# AUTOMATIC LOCAL ANNEALING

Jared Leinbach

Department of Psychology

Carnegie-Mellon University

Pittsburgh, PA 15213

## ABSTRACT

This research involves a method for finding global maxima in constraint satisfaction networks. It is an annealing process but, unlike most others, requires no annealing schedule. Temperature is instead determined locally by units at each update, and thus *all* processing is done at the unit level. There are two major practical benefits to processing this way: 1) processing can continue in 'bad' areas of the network, while 'good' areas remain stable, and 2) processing continues in the 'bad' areas, as long as the constraints remain poorly satisfied (i.e. it does not stop after some predetermined number of cycles). As a result, this method not only avoids the kludge of requiring an externally determined annealing schedule, but it also finds global maxima more quickly and consistently than externally scheduled systems (a comparison to the Boltzmann machine (Ackley et al, 1985) is made). Finally, implementation of this method is computationally trivial.

## INTRODUCTION

A constraint satisfaction network, is a network whose units represent hypotheses, between which there are various constraints. These constraints are represented by bi-directional connections between the units. A positive connection weight suggests that if one hypothesis is accepted or rejected, the other one should be also, and a negative connection weight suggests that if one hypothesis is accepted or rejected, the other one should not be. The relative importance of satisfying each constraint is indicated by the absolute size of the corresponding weight. The acceptance or rejection of a hypothesis is indicated by the activation of the corresponding unit. Thus every point in the activation space corresponds to a possible solution to the constraint problem represented by the network. The quality of any solution can be calculated by summing the 'satisfiedness' of all the constraints. The goal is to find a point in the activation space for which the quality is at a maximum.

Unfortunately, if units update deterministically (i.e. if they *always* move toward the state that best satisfies their constraints) there is no means of avoiding local quality maxima in the activation space. This is simply a fundamental problem of all gradient decent procedures. Annealing systems attempt to avoid this problem by always giving units some probability of *not* moving towards the state that best satisfies their constraints. This probability is called the 'temperature' of the network. When the temperature is high, solutions are generally not good, but the network moves easily throughout the activation space. When the temperature is low, the network is committed to one area of the activation space, but it is very good at improving its solution within that area. Thus the annealing analogy is born. The notion is that if you start with the temperature high, and lower it slowly enough, the network will gradually replace its 'state mobility' with 'state improvement ability', in such a way as to guide itself into a globally maximal state (much as the atoms in slowly annealed metals find optimal bonding structures).

To search for solutions this way, requires some means of determining a temperature for the network, at every update. Annealing systems simply use a predetermined schedule to provide this information. However, there are both practical and theoretical problems with this approach. The main practical problems are the following: 1) once an annealing schedule comes to an end, all processing is finished regardless of the quality of the current solution, and 2) temperature must be uniform across the network, even though different parts of the network may merit different temperatures (this is the case any time one part of the network is in a 'better' area of the activation space than another, which is a natural condition). The theoretical problem with this approach involves the selection of annealing schedules. In order to pick an appropriate schedule for a network, one *must* use some knowledge about what a good solution for that network is. Thus in order to get the system to find a solution, you must already know something about the solution you want it to find. The problem is that one of the most critical elements of the process, the way that the temperature is decreased, is handled by something other than the network itself. Thus the quality of the final solution must depend, at least in part, on *that* system's understanding of the problem.

By allowing each unit to control its own temperature during processing, Automatic Local Annealing avoids this serious kludge. In addition, by resolving the main practical problems, it also ends up finding global maxima more quickly and reliably than externally controlled systems.

## MECHANICS

All units take on continuous activations between a uniform minimum and maximum value. There is also a uniform resting activation for all units (between the minimum and maximum). Units start at random activations, and are updated synchronously at each cycle in one of two possible ways. Either they are updated via any ordinary update rule for which a positive net input (as defined below) increases activation and a negative net input decreases activation, or they are simply reset to their resting activation. There is an update probability function that determines the probability of normal update for a unit based on its temperature (as defined below). It should be noted that once the net input for

a unit has been calculated, finding its temperature is trivial (the quantity $(a_i - rest)$ in the equation for $goodness_i$ can come outside the summation).

**Definitions:**

$$netinput_i \quad = \sum_j (a_j - rest) \times w_{ij}$$

$$temperature_i = \begin{array}{ll} -goodness_i / maxposgdnss_i & \text{if } goodness_i \geq 0 \\ goodness_i / maxneggdnss_i & \text{otherwise} \end{array}$$

$$goodness_i = \sum_j (a_i - rest) \times w_{ij} \times (a_j - rest)$$

$maxposgdnss_i$ = the largest pos. value that $goodness_i$ could be
$maxneggdnss_i$ = the largest neg. value that $goodness_i$ could be

Maxposgdnss and maxneggdnss are constants that can be calculated once for each unit at the beginning of simulation. They depend only on the weights into the unit, and the constant maximum, minimum and resting activation values. Temperature is always a value between 1 and -1, with 1 representing high temperature and -1 low.

# SIMULATIONS

The parameters below were used in processing *both* of the networks that were tested. The first network processed (Figure 1a) has two local maxima that are extremely close to its two global maxima. This is a very 'difficult' network in the sense that the search for a global maximum must be extremely sensitive to the minute difference between the global maxima and the next-best local maxima. The other network processed (Figure 1b) has many local maxima, but none of them are especially close to the global maxima. This is an 'easy' network in the sense that the slow and cautious process that was used, was not really necessary. A more appropriate set of parameters would have improved performance on this second network, but it was not used in order to illustrate the relative generality of the algorithm.

**Parameters:**

maximum activation = 1

minimum activation = 0

resting activation = 0.5

normal update rule:

$$\Delta\,activation_i = \begin{array}{ll} netinput_i \times (maxactivation - activation_i) \times k & \text{if } netinput_i \geq 0 \\ netinput_i \times (activation_i - minactivation) \times k & \text{otherwise} \end{array}$$

with k = 0.6

update probability function:

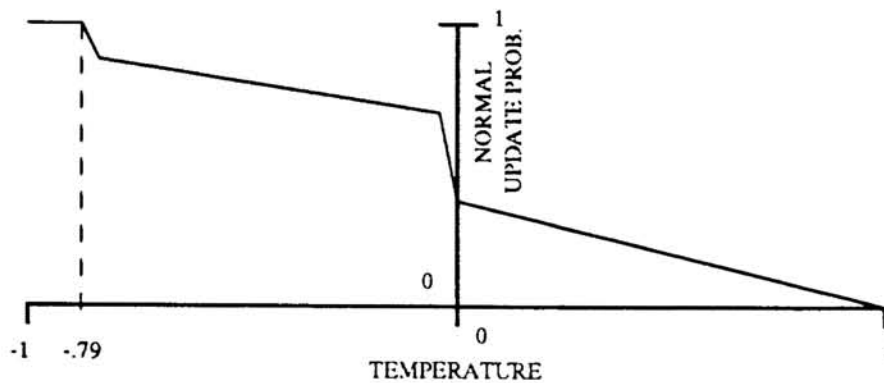

This function defines a process that moves slowly towards a global maximum, moves away from even good solutions easily, and 'freezes' units that are colder than -0.79.

## RESULTS

The results of running the Automatic Local Annealing process on these two networks (in comparison to a standard Boltzmann Machine's performance) are summarized in figures 2a and 2b. With Automatic Local Annealing (ALA), the probability of having found a stable global maximum departs from zero fairly soon after processing begins, and increases smoothly up to one. The Boltzmann Machine, instead, makes little 'useful' progress until the end of the annealing schedule, and then quickly moves into a solution which may or may not be a global maximum. In order to get its reliability near that of ALA, the Boltzmann Machine's schedule must be so slow that solutions are found much more slowly than ALA. Conversely in ordet to start finding solution as quickly as ALA, such a short schedule is necessary that the reliability becomes much worse than ALA's. Finally, if one makes a more reasonable comparison to the Boltzmann Machine (either by changing the parameters of the ALA process to maximize its performance on each network, or by using a single annealing schedule with the Boltzmann Machine for both networks), the overall performance advantage for ALA increases substantially.

## DISCUSSION

## HOW IT WORKS

The characteristics of the approach to a global maximum are determined by the shape of the update probability function. By modifying this shape, one can control such things as: how quickly/steadily the network moves towards a global maximum, how easily it moves away from local maxima, how good a solution must be in order for it to become *completely* stable, and so on. The only critical feature of the function, is that as temperature decreases the probability of normal update increases. In this way, the colder a unit gets the more steadily it progresses towards an extreme activation value, and the hotter a unit gets the more time it spends near resting activation. From this you get hot

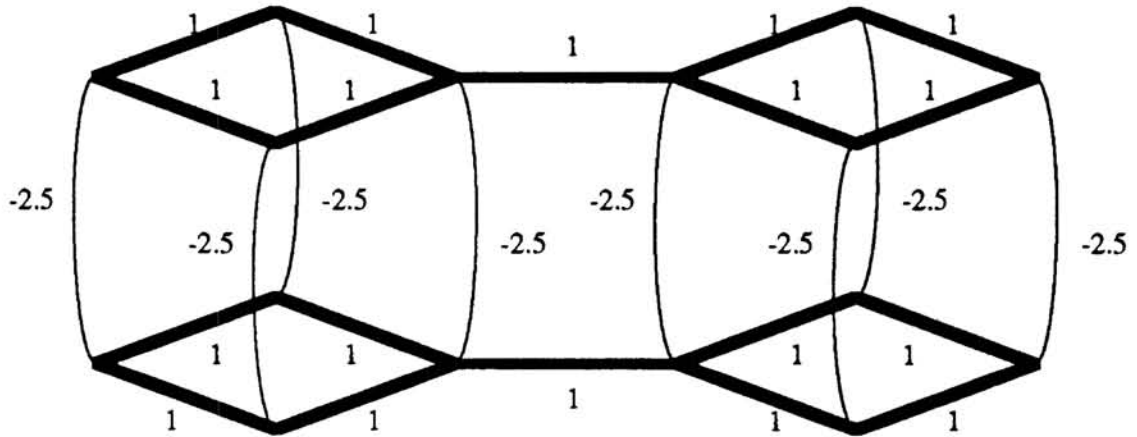

**Figure 1a.** A 'Difficult' Network.

Global maxima are: 1) all eight upper units on, with the remaining units off, 2) all eight lower units on with the remaining units off. Next best local maxima are: 1) four upper left and four lower right units on, with the remaining units off, 2) four upper right and four lower left units on, with the remaining units off.

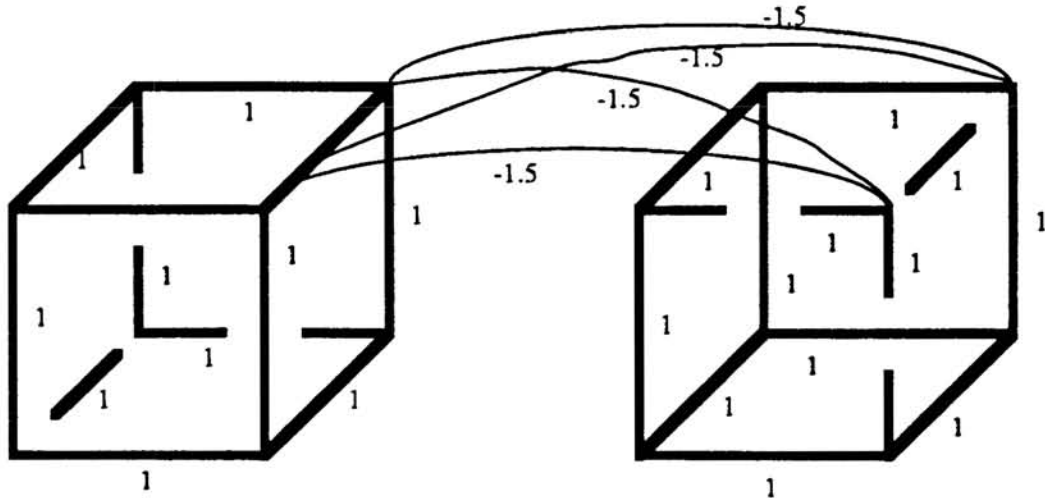

**Figure 1b.** An 'Easy' Network.

Necker cube network (McClelland & Rumelhart 1988). Each set of four corresponding units are connected as shown above. Connections for the other three such sets were omitted for clarity. The global maxima have all units in one cube on with all units in the other off.

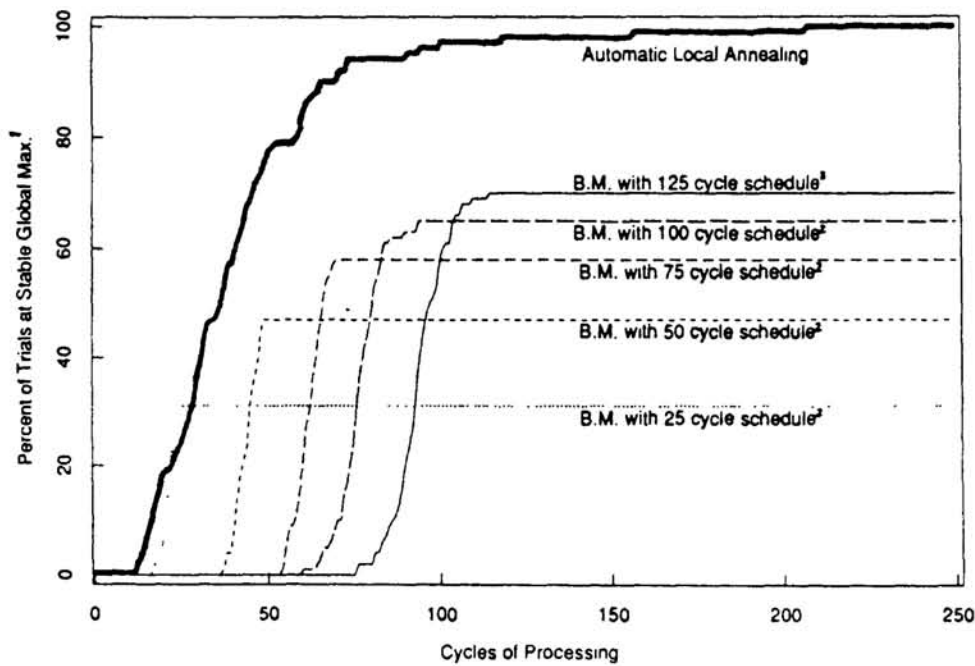

**Figure 2a.** Performance On A 'Difficult' Network (Figure 1a).

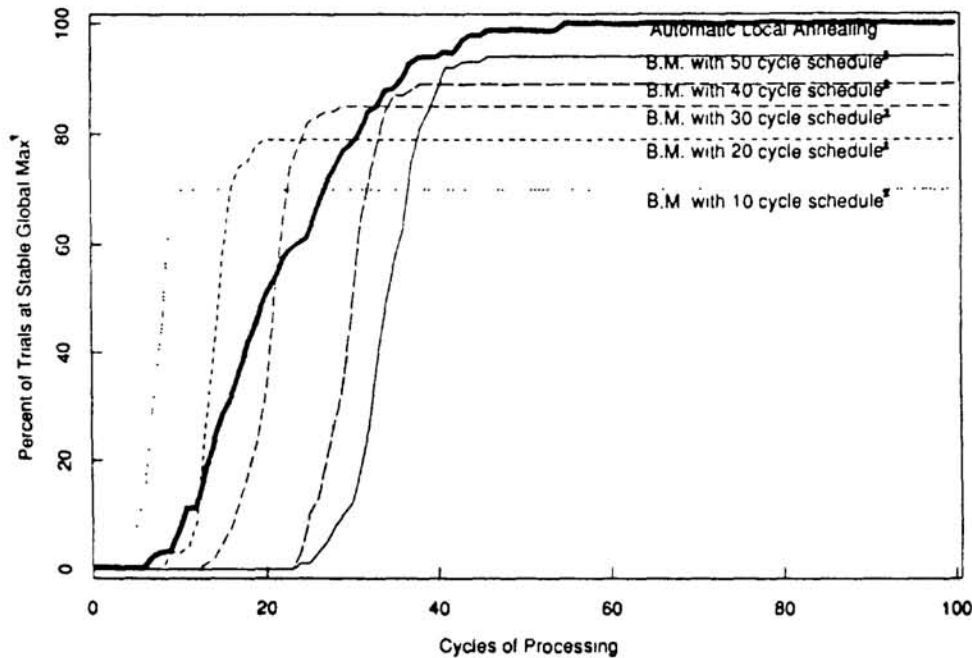

**Figure 2b.** Performance On An 'Easy' Network (Figure 1b).

[1]Each line is based on 100 trials. A *stable* global maxima is one that the network remained in for the rest of the trial.

[2]All annealing schedules were the *best* performing three-leg schedules found.

units that have little effect on movement in the activation space (since they contribute little to any unit's net input), and cold units that compete to control this critical movement.

The cold units 'cool' connected units that are in agreement with them, and 'heat' connected units that are in disagreement (see temperature equation). As the connected agreeing units are cooled, they too begin to cool their connected agreeing units. In this way coldness spreads out, stabilizing sets of units whose hypotheses agree. This spreading is what makes the ALA algorithm work. A units decision about its hypothesis can now be felt by units that are only distantly connected, as must be the case if units are to act in accordance with any *global* criterion (e.g. the overall quality of the states of these networks).

In order to see why global maxima are found, one must consider the network as a whole. In general, the amount of time spent in any state is proportional to the amount of heat in that state (since heat is directly related to stability). The state(s) containing the least possible heat for a given network, will be the most stable. These state(s) will also represent the global maxima (since they have the least total 'dissatisfaction' of constraints). Therefore, given infinite processing time, the most commonly visited states will be the global maxima. More importantly, the 'visitedness' of *every* state will be proportional to its overall quality (a mathematical description of this has not yet been developed).

This later characteristic provides good practical benefits, when one employs a notion of solution satisficing. This is done by using an update probability function that allows units to 'freeze' (i.e. have normal update probabilities of 1) at temperatures higher than -1 (as was done with the simulations described above). In this condition, states can become completely stable, without perfectly satisfying all constraints. As the time of simulation increases, the probability of being in any given state approaches approaches a value proportional to its quality. Thus, if there *are* any states good enough to be frozen, the chances of not having hit one will decrease with time. The amount of time necessary to satisfice is directly related to the freezing point used. Times as small as 0 (for freezing points > 1) and as large as infinity (for freezing points < -1) can be achieved. This type of time/quality trade-off, is extremely useful in many practical applications.

## MEASURING PERFORMANCE

While ALA finds global maxima faster and more reliably than Boltzmann Machine annealing, these are not the only benefits to ALA processing. A number of other elements make it preferable to externally scheduled annealing processes: 1) Various solutions to subparts of problems are found and, at least temporarily, maintained during processing. If one considers constraint satisfaction networks in terms of schema processors, this corresponds nicely to the simultaneous processing of all levels of schemas and subschemas. Subschemas with obvious solutions get filled in quickly, even when the higher level schemas have still not found real solutions. While these initial sub-solutions may not end up as part of the final solution, their appearance during

processing can still be quite useful in some settings. 2) ALA is much more biologically feasible than externally scheduled systems. Not only can units function on their own (without the use of an intelligent external processor), but the paths traversed through the activation space (as described by the schema example above) also parallel human processing more closely. 3) ALA processing may lend itself to simple learning algorithms. During processing, units are always acting in close accord with the constraints that are present. At first distant constraint are ignored in favor of more immediate ones, but regardless the units rarely actually *defy* any constraints in the network. Thus basic approaches to making weight adjustments, such as continuously increasing weights between units that are in agreement about their hypotheses, and decreasing weights between units that are in disagreement about their hypotheses (Minsky & Papert, 1968), may have new power. This is an area of current research, which would represent an enormous time savings over Boltzmann Machine type learning (Ackley et al 1985) if it were to be found feasible.

# REFERENCES

Ackley, D. H., Hinton, G. E., & Sejnowski, T. J. (1985). A Learning Algorithm for Boltzmann Machines. *Cognitive Science*, 9, 147-169.

McClelland, J. L., & Rumelhart, D. E. (1988). *Explorations in Parallel Distributed Processing*. Cambridge, MA: MIT Press.

Minsky, M., & Papert, S. (1968). *Perceptrons*. Cambridge, MA: MIT Press.
